# Chained Boosting

**Christian R. Shelton**
University of California
Riverside CA 92521
cshelton@cs.ucr.edu

**Wesley Huie**
University of California
Riverside CA 92521
whuie@cs.ucr.edu

**Kin Fai Kan**
University of California
Riverside CA 92521
kkan@cs.ucr.edu

## Abstract

We describe a method to learn to make sequential stopping decisions, such as those made along a processing pipeline. We envision a scenario in which a series of decisions must be made as to whether to continue to process. Further processing costs time and resources, but may add value. Our goal is to create, based on historic data, a series of decision rules (one at each stage in the pipeline) that decide, based on information gathered up to that point, whether to continue processing the part. We demonstrate how our framework encompasses problems from manufacturing to vision processing. We derive a quadratic (in the number of decisions) bound on testing performance and provide empirical results on object detection.

## 1 Pipelined Decisions

In many decision problems, all of the data do not arrive at the same time. Often further data collection can be expensive and we would like to make a decision without accruing the added cost.

Consider silicon wafer manufacturing. The wafer is processed in a series of stages. After each stage some tests are performed to judge the quality of the wafer. If the wafer fails (due to flaws), then the processing time, energy, and materials are wasted. So, we would like to detect such a failure as early as possible in the production pipeline.

A similar problem can occur in vision processing. Consider the case of object detection in images. Often low-level pixel operations (such as downsampling an image) can be performed in parallel by dedicated hardware (on a video capture board, for example). However, searching each subimage patch of the whole image to test whether it is the object in question takes time that is proportional to the number of pixels. Therefore, we can imagine a image pipeline in which low resolution versions of the whole image are scanned first. Subimages which are extremely unlikely to contain the desired object are rejected and only those which pass are processed at higher resolution. In this way, we save on many pixel operations and can reduce the cost in time to process an image.

Even if downsampling is not possible through dedicated hardware, for most object detection schemes, the image must be downsampled to form an image pyramid in order to search for the object at different scales. Therefore, we can run the early stages of such a pipelined detector at the low resolution versions of the image and throw out large regions of the high resolution versions. Most of the processing is spent searching for small faces (at the high resolutions), so this method can save a lot of processing.

Such chained decisions also occur if there is a human in the decision process (to ask further clarifying questions in database search, for instance). We propose a framework that can model all of these scenarios and allow such decision rules to be learned from historic data. We give a learning algorithm based on the minimization of the exponential loss and conclude with some experimental results.

### 1.1  Problem Formulation

Let there be $s$ stages to the processing pipeline. We assume that there is a static distribution from which the parts, objects, or units to be processed are drawn. Let $p(x, c)$ represent this distribution in which $x$ is a vector of the features of this unit and $c$ represents the costs associated with this unit. In particular, let $x_i$ ($1 \leq i \leq s$) be the set of measurements (features) available to the decision maker immediately following stage $i$. Let $c_i$ ($1 \leq i \leq s$) be the cost of rejecting (or stopping the processing of) this unit immediately following stage $i$. Finally, let $c_{s+1}$ be the cost of allowing the part to pass through all processing stages.

Note that $c_i$ need not be monotonic in $i$. To take our wafer manufacturing example, for wafers that are good we might let $c_i = i$ for $1 \leq i \leq s$, indicating that if a wafer is rejected at any stage, one unit of work has been invested for each stage of processing. For the same good wafers, we might let $c_{s+1} = s - 1000$, indicating that the value of a completed wafer is 1000 units and therefore the total cost is the processing cost minus the resulting value. For a flawed wafer, the values might be the same, except for $c_{s+1}$ which we would set to $s$, indicating that there is no value for a bad wafer.

Note that the costs may be either positive or negative. However, only their relative values are important. Once a part has been drawn from the distribution, there is no way of affecting the "base level" for the value of the part. Therefore, we assume for the remainder of this paper that $c_i \geq 0$ for $1 \leq i \leq s+1$ and that $c_i = 0$ for some value of $i$ (between 1 and $s+1$).

Our goal is to produce a series of decision rules $f_i(x_i)$ for $1 \leq i \leq s$. We let $f_i$ have a range of $\{0, 1\}$ and let 0 indicate that processing should continue and 1 indicate that processing should be halted. We let $f$ denote the collection of these $s$ decision rules and augment the collection with an additional rule $f_{s+1}$ which is identically 1 (for ease of notation). The cost of using these rules to halt processing an example is therefore

$$L(f(x), c) = \sum_{i=1}^{s+1} c_i f_i(x_i) \prod_{j=1}^{i-1} (1 - f_j(x_j)) \ .$$

We would like to find a set of decision rules that minimize $E_p[L(f(x), c)]$.

While $p(x, c)$ is not known, we do have a series of samples (training set) $\mathcal{D} = \{(x^1, c^1), (x^2, c^2), \dots, (x^n, c^n)\}$ of $n$ examples drawn from the distribution $p$. We use superscripts to denote the example index and subscripts to denote the stage index.

## 2  Boosting Solution

For this paper, we consider constructing the rules $f_i$ from simpler decision rules, much as in the Adaboost algorithm [1, 2]. We assume that each decision $f_i(x_i)$ is computed as the threshold of another function $g_i(x_i)$: $f_i(x_i) = \mathcal{I}(g_i(x_i) > 0)$.[1] We bound the empirical risk:

$$\sum_{k=1}^{n} L(f(x^k), c^k) = \sum_{k=1}^{n} \sum_{i=1}^{s+1} c_i^k \mathcal{I}(g_i(x_i^k) > 0) \prod_{j=1}^{i-1} \mathcal{I}(g_j(x_j^k) \leq 0)$$

$$\leq \sum_{k=1}^{n} \sum_{i=1}^{s+1} c_i^k e^{g_i(x_i^k)} \prod_{j=1}^{i-1} e^{-g_j(x_j^k)} = \sum_{k=1}^{n} \sum_{i=1}^{s+1} c_i^k e^{g_i(x_i^k) - \sum_{j=1}^{i-1} g_j(x_j^k)} \ . \quad (1)$$

Our decision to make all costs positive ensures that the bounds hold. Our decision to make the optimal cost zero helps to ensure that the bound is reasonably tight.

As in boosting, we restrict $g_i(x_i)$ to take the form $\sum_{l=1}^{m_i} \alpha_{i,l} h_{i,l}(x_i)$, the weighted sum of $m_i$ sub-classifiers, each of which returns either $-1$ or $+1$. We will construct these weighted sums incrementally and greedily, adding one additional subclassifier and associated weight at each step. We will pick the stage, weight, and function of the subclassifier in order to make the largest negative change in the exponential bound to the empirical risk. The subclassifiers, $h_{i,l}$ will be drawn from a small class of hypotheses, $\mathcal{H}$.

1. Initialize $g_i(x) = 0$ for all stages $i$
2. Initialize $w_i^k = c_i^k$ for all stages $i$ and examples $k$.
3. For each stage $i$:
   (a) Calculate targets for each training example, as shown in equation 5.
   (b) Let $h$ be the result of running the base learner on this set.
   (c) Calculate the corresponding $\alpha$ as per equation 3.
   (d) Score this classification as per equation 4
4. Select the stage $\bar{\imath}$ with the best (highest) score. Let $\bar{h}$ and $\bar{\alpha}$ be the classifier and weight found at that stage.
5. Let $g_{\bar{\imath}}(x) \leftarrow g_{\bar{\imath}}(x) + \bar{\alpha}\bar{h}(x)$.
6. Update the weights (see equation 2):
   - $\forall 1 \le k \le n$, multiply $w_{\bar{\imath}}^k$ by $e^{\bar{\alpha}\bar{h}(x_{\bar{\imath}}^k)}$.
   - $\forall 1 \le k \le n, j > \bar{\imath}$, multiply $w_j^k$ by $e^{-\bar{\alpha}\bar{h}(x_{\bar{\imath}}^k)}$.
7. Repeat from step 3

Figure 1: Chained Boosting Algorithm

## 2.1 Weight Optimization

We first assume that the stage at which to add a new subclassifier and the subclassifier to add have already been chosen: $\bar{\imath}$ and $\bar{h}$, respectively. That is, $\bar{h}$ will become $h_{\bar{\imath},m_{\bar{\imath}}+1}$ but we simplify it for ease of expression. Our goal is to find $\alpha_{\bar{\imath},m_{\bar{\imath}}+1}$ which we similarly abbreviate to $\bar{\alpha}$. We first define

$$w_i^k = c_i^k e^{g_i(x_i^k) - \sum_{j=1}^{i-1} g_j(x_j^k)} \tag{2}$$

as the weight of example $k$ at stage $i$, or its current contribution to our risk bound. If we let $\mathcal{D}_{\bar{h}}^+$ be the set of indexes of the members of $\mathcal{D}$ for which $\bar{h}$ returns $+1$, and let $\mathcal{D}_{\bar{h}}^-$ be similarly defined for those for which $\bar{h}$ returns $-1$, we can further define

$$W_{\bar{\imath}}^+ = \sum_{k \in \mathcal{D}_{\bar{h}}^+} w_{\bar{\imath}}^k + \sum_{k \in \mathcal{D}_{\bar{h}}^-} \sum_{i=\bar{\imath}+1}^{s+1} w_i^k \qquad W_{\bar{\imath}}^- = \sum_{k \in \mathcal{D}_{\bar{h}}^-} w_{\bar{\imath}}^k + \sum_{k \in \mathcal{D}_{\bar{h}}^+} \sum_{i=\bar{\imath}+1}^{s+1} w_i^k \ .$$

We interpret $W_{\bar{\imath}}^+$ to be the sum of the weights which $\bar{h}$ will emphasize. That is, it corresponds to the weights along the path that $\bar{h}$ selects: For those examples for which $\bar{h}$ recommends termination, we add the current weight (related to the cost of stopping the processing at this stage). For those examples for which $\bar{h}$ recommends continued processing, we add in all future weights (related to all future costs associated with this example). $W_{\bar{\imath}}^-$ can be similarly interpreted to be the weights (or costs) that $\bar{h}$ recommends skipping.

If we optimize the loss bound of Equation 1 with respect to $\bar{\alpha}$, we obtain

$$\bar{\alpha} = \frac{1}{2} \log \frac{W_{\bar{\imath}}^-}{W_{\bar{\imath}}^+} \ . \tag{3}$$

The more weight (cost) that the rule recommends to skip, the higher its $\alpha$ coefficient.

## 2.2 Full Optimization

Using Equation 3 it is straight forward to show that the reduction in Equation 1 due to the addition of this new subclassifier will be

$$W_{\bar{\imath}}^+(1 - e^{\bar{\alpha}}) + W_{\bar{\imath}}^-(1 - e^{-\bar{\alpha}}) \ . \tag{4}$$

We know of no efficient method for determining $\bar{\imath}$, the stage at which to add a subclassifier, except by exhaustive search. However, within a stage, the choice of which subclassifier to use becomes one

of maximizing

$$\sum_{k=1}^{n} z_{\bar{i}}^{k} \bar{h}(x_{\bar{i}}^{k}) \ , \ \ \text{where} \ \ z_{\bar{i}}^{k} = \left[\sum_{i=\bar{i}+1}^{s+1} w_i^k\right] - w_{\bar{i}}^k \tag{5}$$

with respect to $\bar{h}$. This is equivalent to an weighted empirical risk minimization where the training set is $\{x_{\bar{i}}^1, x_{\bar{i}}^2, \ldots, x_{\bar{i}}^n\}$. The label of $x_{\bar{i}}^k$ is the sign of $z_{\bar{i}}^k$, and the weight of the same example is the magnitude of $z_{\bar{i}}^k$.

## 2.3   Algorithm

The resulting algorithm is only slightly more complex than standard Adaboost. Instead of a weight vector (one weight for each data example), we now have a weight matrix (one weight for each data example for each stage). We initialize each weight to be the cost associated with halting the corresponding example at the corresponding stage. We start with all $g_i(x) = 0$. The complete algorithm is as in Figure 1.

Each time through steps 3 through 7, we complete one "round" and add one additional rule to one stage of the processing. We stop executing this loop when $\bar{\alpha} \leq 0$ or when an iteration counter exceeds a preset threshold.

### Bottom-Up Variation

In situations where information is only gained after each stage (such as in section 4), we can also train the classifiers "bottom-up." That is, we can start by only adding classifiers to the last stage. Once finished with it, we proceed to the previous stage, and so on. Thus instead of selecting the best stage, $i$, in each round, we systematically work our way backward through the stages, never revisiting previously set stages.

## 3   Performance Bounds

Using the bounds in [3] we can provide a risk bound for this problem. We let $E$ denote the expectation with respect to the true distribution $p(x, c)$ and $\hat{E}_n$ denote the empirical average with respect to the $n$ training samples. We first bound the indicator function with a piece-wise linear function, $b_\theta$, with a maximum slope of $\frac{1}{\theta}$:

$$\mathcal{I}(z > 0) \leq b_\theta(z) = \max\left(\min\left(1, 1 + \frac{z}{\theta}\right), 0\right) \ .$$

We then bound the loss: $L(f(x), c) \leq \phi_\theta(f(x), c)$ where

$$\phi_\theta(f(x), c) = \sum_{i=1}^{s+1} c_i \min\{b_\theta(g_i(x_i)), b_\theta(-g_{i-1}(x_{i-1})), b_\theta(-g_{i-2}(x_{i-2})), \ldots, b_\theta(-g_1(x_1))\}$$

$$= \sum_{i=1}^{s+1} c_i B_\theta^i(g_i(x_i), g_{i-1}(x_{i-1}), \ldots, g_1(x_1))$$

We replaced the product of indicator functions with a minimization and then bounded each indicator with $b_\theta$. $B_\theta^i$ is just a more compact presentation of the composition of the function $b_\theta$ and the minimization. We assume that the weights $\alpha$ at each stage have been scaled to sum to 1. This has no affect on the resulting classifications, but is necessary for the derivation below. Before stating the theorem, for clarity, we state two standard definition:

**Definition 1.** *Let $p(x)$ be a probability distribution on the set $\mathcal{X}$ and let $\{x^1, x^2, \ldots, x^n\}$ be $n$ independent samples from $p(x)$. Let $\sigma^1, \sigma^2, \ldots, \sigma^n$ be $n$ independent samples from a Rademacher random variable (a binary variable that takes on either $+1$ or $-1$ with equal probability). Let $\mathcal{F}$ be a class of functions mapping $\mathcal{X}$ to $\Re$.*

*Define the* Rademacher Complexity *of $\mathcal{F}$ to be*

$$R_n(\mathcal{F}) = E\left[\sup_{f \in \mathcal{F}} \frac{1}{n} \left|\sum_{i=1}^{n} \sigma^i f(x^i)\right|\right]$$

*where the expectation is over the random draws of $x^1$ through $x^n$ and $\sigma^1$ through $\sigma^n$.*

**Definition 2.** *Let $p(x)$, $\{x^1, x^2, \ldots, x^n\}$, and $\mathcal{F}$ be as above. Let $g^1, g^2, \ldots, g^n$ be $n$ independent samples from a Gaussian distribution with mean 0 and variance 1.*

*Analogous to the above definition, define the* Gaussian Complexity *of $\mathcal{G}$ to be*

$$G_n(\mathcal{F}) = E\left[\sup_{f \in \mathcal{F}} \frac{1}{n} \left| \sum_{i=1}^{n} g^i f(x^i) \right| \right] .$$

We can now state our theorem, bounding the true risk by a function of the empirical risk:

**Theorem 3.** *Let $\mathcal{H}_1, \mathcal{H}_2, \ldots, \mathcal{H}_s$ be the sequence of the sets of functions from which the base classifier draws for chain boosting. If $\mathcal{H}_i$ is closed under negation for all $i$, all costs are bounded between 0 and 1, and the weights for the classifiers at each stage sum to 1, then with probability $1 - \delta$,*

$$E\left[L(f(x), c)\right] \le \hat{E}_n\left[\phi_\theta(f(x), c)\right] + \frac{k}{\theta} \sum_{i=1}^{s} (i+1) G_n(\mathcal{H}_i) + \sqrt{\frac{8 \ln \frac{2}{\delta}}{n}}$$

*for some constant k.*

*Proof.* Theorem 8 of [3] states

$$E\left[L(x, c)\right] \le \hat{E}_n\left(\phi_\theta(f(x), c)\right) + 2R_n(\phi_\theta \circ \mathcal{F}) + \sqrt{\frac{8 \ln \frac{2}{\delta}}{n}}$$

and therefore we need only bound the $R_n(\phi_\theta \circ \mathcal{F})$ term to demonstrate our theorem. For our case, we have

$$
\begin{aligned}
R_n(\phi_\theta \circ \mathcal{F}) &= E \sup_{f \in \mathcal{F}} \frac{1}{n} \left| \sum_{i=1}^{n} \sigma^i \phi_\theta(f(x^i), c^i) \right| \\
&= E \sup_{f \in \mathcal{F}} \frac{1}{n} \left| \sum_{i=1}^{n} \sigma^i \sum_{j=1}^{s+1} c_j^i B_\theta^s(g_j(x_j^i), g_{j-1}(x_{j-1}^i), \ldots, g_1(x_1^i)) \right| \\
&\le \sum_{j=1}^{s+1} E \sup_{f \in \mathcal{F}} \frac{1}{n} \left| \sum_{i=1}^{n} \sigma^i B_\theta^s(g_j(x_j^i), g_{j-1}(x_{j-1}^i), \ldots, g_1(x_1^i)) \right| = \sum_{j=1}^{s+1} R_n(B_\theta^s \circ \mathcal{G}^j)
\end{aligned}
$$

where $\mathcal{G}_i$ is the space of convex combinations of functions from $\mathcal{H}_i$ and $\mathcal{G}^i$ is the cross product of $\mathcal{G}_1$ through $\mathcal{G}_i$. The inequality comes from switching the expectation and the maximization and then from dropping the $c_j^i$ (see [4], lemma 5).

Lemma 4 of [3] states that there exists a $k$ such that $R_n(B_\theta^s \circ \mathcal{G}^j) \le k G_n(B_\theta^s \circ \mathcal{G}^j)$. Theorem 14 of the same paper allows us to conclude that $G_n(B_\theta^s \circ \mathcal{G}^j) \le \frac{2}{\theta} \sum_{i=1}^{j} G_n(\mathcal{G}_i)$. (Because $B_\theta^s$ is the minimum over a set of functions with maximum slope of $\frac{1}{\theta}$, the maximum slope of $B_\theta^s$ is also $\frac{1}{\theta}$.) Theorem 12, part 2 states $G_n(\mathcal{G}_i) = G_n(\mathcal{H}_i)$. Taken together, this proves our result. $\square$

Note that this bound has only quadratic dependence on $s$, the length of the chain and does not explicitly depend on the number of rounds of boosting (the number of rounds affects $\phi_\theta$ which, in turn, affects the bound).

## 4 Application

We tested our algorithm on the MIT face database [5]. This database contains 19-by-19 gray-scale images of faces and non-faces. The training set has 2429 face images and 4548 non-face images. The testing set has 472 faces and 23573 non-faces. We weighted the training set images so that the ratio of the weight of face images to non-face images matched the ratio in the testing set.

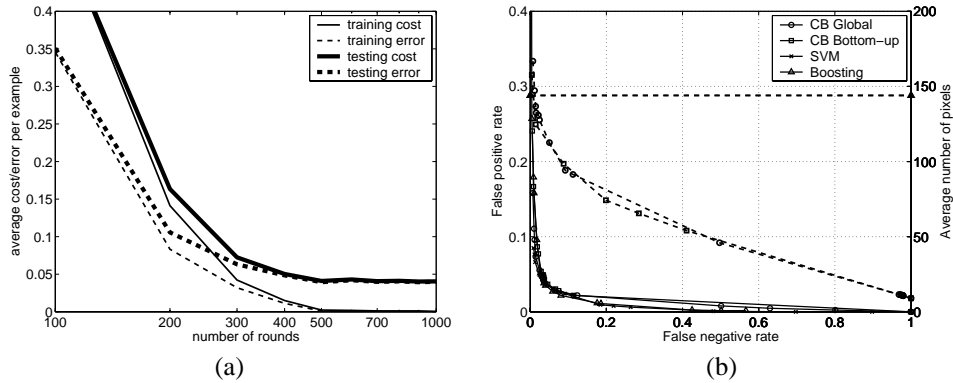

Figure 2: (a) Accuracy verses the number of rounds for a typical run, (b) Error rates and average costs for a variety of cost settings.

## 4.1 Object Detection as Chained Boosting

Our goal is to produce a classifier that can identify non-face images at very low resolutions, thereby allowing for quick processing of large images (as explained later). Most image patches (or sub-windows) do not contain faces. We, therefore, built a multi-stage detection system where any early rejection is labeled as a non-face. The first stage looks at image patches of size 3-by-3 (*i.e.* a lower-resolution version of the 19-by-19 original image). The next stage looks at the same image, but at a resolution of 6-by-6. The third stage considers the image at 12-by-12. We did not present the full 19-by-19 images as the classification did not significantly improve over the 12-by-12 versions.

We employ a simple base classifier: the set of all functions that look at a single pixel and predict the class by thresholding the pixel's value. The total classifier at any stage is a linear combination of these simple classifiers. For a given stage, all of the base classifiers that target a particular pixel are added together producing a complex function of the value of the pixel. Yet, this pixel can only take on a finite number of values (256 in this case). Therefore, we can compile this set of base classifiers into a single look-up function that maps the brightness of the pixel into a real number. The total classifier for the whole stage is merely the sum of these look-up functions. Therefore, the total work necessary to compute the classification at a stage is proportional to the number of pixels in the image considered at that stage, regardless of the number of base classifiers used.

We therefore assign a cost to each stage of processing proportional to the number of pixels at that stage. If the image is a face, we add a negative cost (*i.e.* bonus) if the image is allowed to pass through all of the processing stages (and is therefore "accepted" as a face). If the image is a non-face, we add a bonus if the image is rejected at any stage before completion (*i.e.* correctly labelled).

While this dataset has only segmented image patches, in a real application, the classifier would be run on all sub-windows of an image. More importantly, it would also be run at multiple resolutions in order to detect faces of different sizes (or at different distances from the camera). The classifier chain could be run simultaneously at each of these resolutions. To wit, while running the final 12-by-12 stage at one resolution of the image, the 6-by-6 (previous) stage could be run at the same image resolution. This 6-by-6 processing would be the necessary pre-processing step to running the 12-by-12 stage at a higher resolution. As we run our final scan for big faces (at a low resolution), we can already (at the same image resolution) be performing initial tests to throw out portions of the image as not worthy of testing for smaller faces (at a higher resolution). Most of the work of detecting objects must be done at the high resolutions because there are many more overlapping subwindows. This chained method allows the culling of most of this high-resolution image processing.

## 4.2 Experiments

For each example, we construct a vector of stage costs as above. We add a constant to this vector to ensure that the minimal element is zero, as per section 1.1. We scale all vectors by the same amount

to ensure that the maximal value is 1.This means that the number of misclassifications is an upper bound on the total cost that the learning algorithm is trying to minimize.

There are three flexible quantities in this problem formulation: the cost of a pixel evaluation, the bonus for a correct face classification, and the bonus for a correct non-face classification. Changing these quantities will control the trade-off between false positives and true positives, and between classification error and speed.

Figure 2(a) shows the result of a typical run of the algorithm. As a function of the number of rounds, it plots the cost (that which the algorithm is trying to minimize) and the error (number of misclassified image patches), for both the training and testing sets (where the training set has been reweighted to have the same proportion of faces to non-faces as the testing set).

We compared our algorithm's performance to the performance of support vector machines (SVM) [6] and Adaboost [1] trained and tested on the highest resolution, 12-by-12, image patches. We employed SVM-light [7] with a linear kernels. Figure 2(b) compares the error rates for the methods (solid lines, read against the left vertical axis). Note that the error rates are almost identical for the methods. The dashed lines (read against the right vertical axis) show the average number of pixels evaluated (or total processing cost) for each of the methods. The SVM and Adaboost algorithms have a constant processing cost. Our method (by either training scheme) produces lower processing cost for most error rates.

## 5  Related Work

Cascade detectors for vision processing (see [8] or [9] for example) may appear to be similar to the work in this paper. Especially at first glance for the area of object detection, they appear almost the same. However, cascade detection and this work (chained detection) are quite different.

Cascade detectors are built one at a time. A coarse detector is first trained. The examples which pass that detector are then passed to a finer detector for training, and so on. A series of targets for false-positive rates define the increasing accuracy of the detector cascade.

By contrast, our chain detectors are trained as an ensemble. This is necessary because of two differences in the problem formulation. First, we assume that the information available at each stage changes. Second, we assume there is an explicit cost model that dictates the cost of proceeding from stage to stage and the cost of rejection (or acceptance) at any particular stage. By contrast, cascade detectors are seeking to minimize computational power necessary for a fixed decision. Therefore, the information available to all of the stages is the same, and there are no fixed costs associated with each stage.

The ability to train all of the classifiers at the same time is crucial to good performance in our framework. The first classifier in the chain cannot determine whether it is advantageous to send an example further along unless it knows how the later stages will process the example. Conversely, the later stages cannot construct optimal classifications until they know the distribution of examples that they will see.

Section 4.1 may further confuse the matter. We demonstrated how chained boosting can be used to reduce the computational costs of object detection in images. Cascade detectors are often used for the same purpose. However, the reductions in computational time come from two different sources. In cascade detectors, the time taken to evaluate a given image patch is reduced. In our chained detector formulation, image patches are ignored completely based on analysis of lower resolution patches in the image pyramid. To further illustrate the difference, cascade detectors can always be used to speed up asymmetric classification tasks (and are often applied to image detection). By contrast, in Section 4.1 we have exploited the fact that object detection in images is typically performed at multiple scales to turn the problem into a pipeline and apply our framework.

Cascade detectors address situations in which prior class probabilities are not equal, while chained detectors address situations in which information is gained at a cost. Both are valid (and separate) ways of tackling image processing (and other tasks as well). In many ways, they are complementary approaches.

Classic sequence analysis [10, 11] also addresses the problem of optimal stopping. However, it assumes that the samples are drawn *i.i.d.* from (usually) a known distribution. Our problem is quite different in that each consecutive sample is drawn from a different (and related) distribution and our goal is to find a decision rule without producing a generative model. WaldBoost [12] is a boosting algorithm based on this. It builds a series of features and a ratio comparison test in order to decide when to stop. For WaldBoost, the available features (information) not change between stages. Rather, any feature is available for selection at any point in the chain. Again, this is a different problem than the one considered in this paper.

## 6 Conclusions

We feel this framework of staged decision making is useful in a wide variety of areas. This paper demonstrated how the framework applies to one vision processing task. Obviously it also applies to manufacturing pipelines where errors can be introduced at different stages. It should also be applicable to scenarios where information gathering is costly.

Our current formulation only allows for early negative detection. In the face detection example above, this means that in order to report "face," the classifier must process each stage, even if the result is assured earlier. In Figure 2(b), clearly the upper-left corner (100% false positives and 0% false negatives) is reachable with little effort: classify everything positive without looking at any features. We would like to extend this framework to cover such two-sided early decisions. While perhaps not useful in manufacturing (or even face detection, where the interesting part of the ROC curve is far from the upper-left), it would make the framework more applicable to information-gathering applications.

**Acknowledgements**

This research was supported through the grant "Adaptive Decision Making for Silicon Wafer Testing" from Intel Research and UC MICRO.

## Footnotes

[1] $\mathcal{I}$ is the indicator function that equals 1 if the argument is true and 0 otherwise.

## References

[1] Yoav Freund and Robert E. Schapire. A decision-theoretic generalization of on-line learning and an application to boosting. In *EuroCOLT*, pages 23–37, 1995.

[2] Yoav Freund and Robert E. Schapire. Experiments with a new boosting algorithm. In *ICML*, pages 148–156, 1996.

[3] Peter L. Bartlett and Shahar Mendelson. Rademacher and Gaussian complexities: Risk bounds and structural results. *JMLR*, 2:463–482, 2002.

[4] Ron Meir and Tong Zhang. Generalization error bounds for Bayesian mixture algorithms. *JMLR*, 4:839–860, 2003.

[5] MIT. CBCL face database #1, 2000. http://cbcl.mit.edu/cbcl/software-datasets/FaceData2.html.

[6] Bernhard E. Boser, Isabelle M. Guyon, and Vladimir N. Vapnik. A training algorithm for optimal margin classifiers. In *COLT*, pages 144–152, 1992.

[7] T. Joachims. Making large-scale SVM learning practical. In B. Schlkopf, C. Burges, and A. Smola, editors, *Advances in Kernel Methods — Support Vector Learning*. MIT-Press, 1999.

[8] Paul A. Viola and Michael J. Jones. Rapid object detection using a boosted cascade of simple features. In *CVPR*, pages 511–518, 2001.

[9] Jianxin Wu, Matthew D. Mullin, and James M. Rehg. Linear asymmetric classifier for cascade detectors. In *ICML*, pages 988–995, 2005.

[10] Abraham Wald. *Sequential Analysis*. Chapman & Hall, Ltd., 1947.

[11] K. S. Fu. *Sequential Methods in Pattern Recognition and Machine Learning*. Academic Press, 1968.

[12] Jan Šochman and Jiří Matas. Waldboost — learning for time constrained sequential detection. In *CVPR*, pages 150–156, 2005.
